# Structured output regression for detection with partial truncation

**Andrea Vedaldi**     **Andrew Zisserman**
Department of Engineering
University of Oxford
Oxford, UK
{vedaldi,az}@robots.ox.ac.uk

## Abstract

We develop a structured output model for object category detection that explicitly accounts for alignment, multiple aspects and partial truncation in both training and inference. The model is formulated as large margin learning with latent variables and slack rescaling, and both training and inference are computationally efficient.

We make the following contributions: (i) we note that extending the Structured Output Regression formulation of Blaschko and Lampert [1] to include a bias term significantly improves performance; (ii) that alignment (to account for small rotations and anisotropic scalings) can be included as a latent variable and efficiently determined and implemented; (iii) that the latent variable extends to multiple aspects (e.g. left facing, right facing, front) with the same formulation; and (iv), most significantly for performance, that truncated and truncated instances can be included in both training and inference with an explicit truncation mask.

We demonstrate the method by training and testing on the PASCAL VOC 2007 data set – training includes the truncated examples, and in testing object instances are detected at multiple scales, alignments, and with significant truncations.

## 1   Introduction

There has been a steady increase in the performance of object category detection as measured by the annual PASCAL VOC challenges [3]. The training data provided for these challenges specifies if an object is truncated – when the provided axis aligned bounding box does not cover the full extent of the object. The principal cause of truncation is that the object partially lies outside the image area. Most participants simple disregard the truncated training instances and learn from the non-truncated ones. This is a waste of training material, but more seriously many truncated instances are missed in testing, significantly reducing the recall and hence decreasing overall recognition performance.

In this paper we develop a model (Fig. 1) which explicitly accounts for truncation in both training and testing, and demonstrate that this leads to a significant performance boost. The model is specified as a joint kernel and learnt using an extension of the structural SVM with latent variables framework of [13]. We use this approach as it allows us to address a second deficiency of the provided supervision – that the annotation is limited to axis aligned bounding boxes, even though the objects may be in plane rotated so that the box is a loose bound. The latent variables allow us to specify a pose transformation for each instances so that we achieve a spatial *correspondence* between all instances with the same aspect. We show the benefits of this for both the learnt model and testing performance.

Our model is complementary to that of Felzenszwalb *et al.* [4] who propose a latent SVM framework, where the latent variables specify sub-part locations. The parts give their model some tolerance to in plane rotation and foreshortening (though an axis aligned rectangle is still used for a first

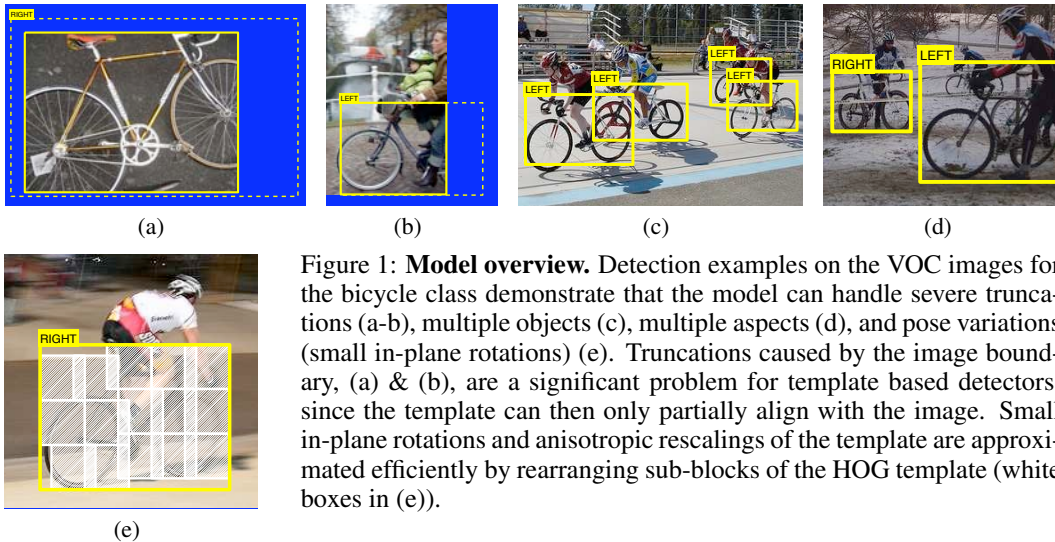

Figure 1: **Model overview.** Detection examples on the VOC images for the bicycle class demonstrate that the model can handle severe truncations (a-b), multiple objects (c), multiple aspects (d), and pose variations (small in-plane rotations) (e). Truncations caused by the image boundary, (a) & (b), are a significant problem for template based detectors, since the template can then only partially align with the image. Small in-plane rotations and anisotropic rescalings of the template are approximated efficiently by rearranging sub-blocks of the HOG template (white boxes in (e)).

stage as a "root filter") but they do not address the problem of truncation. Like them we base our implementation on the efficient and successful HOG descriptor of Dalal and Triggs [2].

Previous authors have also considered occlusion (of which truncation is a special case). Williams *et al.* [11] used pixel wise binary latent variables to specify the occlusion and an Ising prior for spatial coherence. Inference involved marginalizing out the latent variables using a mean field approximation. There was no learning of the model from occluded data. For faces with partial occlusion, the resulting classifier showed an improvement over a classifier which did not model occlusion. Others have explicitly included occlusion at the model learning stage, such as the Constellation model of Fergus *et al.* [5] and the Layout Consistent Random Field model of Winn *et al.* [12]. There are numerous papers on detecting faces with various degrees of partial occlusion from glasses, or synthetic truncations [6, 7].

Our contribution is to define an appropriate joint kernel and loss function to be used in the context of structured output prediction. We then learn a structured regressor, mapping an image to a list of objects with their pose (or bounding box), while at the same time handling explicitly truncation and multiple aspects. Our choice of kernel is inspired by the restriction kernel of [1]; however, our kernel performs both restriction and *alignment* to a template, supports multiple templates to handle different aspects and truncations, and adds a bias term which significantly improves performance.

We refine pose beyond translation and scaling with an additional transformation selected from a finite set of possible perturbations covering aspect ratio change and small in plane rotations. Instead of explicitly transforming the image with each element of this set (which would be prohibitively expensive) we introduce a novel approximation based on decomposing the HOG descriptor into small blocks and quickly rearranging those. To handle occlusions we selectively switch between an object descriptor and an occlusion descriptor. To identify which portions of the template are occluded we use a field of binary variables. These could be treated as latent variables; however, since we consider here only occlusions caused by the image boundaries, we can infer them deterministically from the position of the object relative to the image boundaries. Fig. 1 illustrates various detection examples including truncation, multiple instances and aspects, and in-plane rotation.

In training we improve the ground-truth pose parameters, since the bounding boxes and aspect associations provided in PASCAL VOC are quite coarse indicators of the object pose. For each instance we add a latent variable which encodes a pose adjustment. Such variables are then treated as part of learning using the technique presented in [13]. However, while there the authors use the CCCP algorithm to treat the case of margin rescaling, here we show that a similar algorithm applies to the case of slack rescaling. The resulting optimization alternates between optimizing the model parameters given the latent variables (a convex problem solved by a cutting plane algorithm) and optimizing the latent variable given the model (akin to inference).

The overall method is computationally efficient both in training and testing, achieves very good performances, and it is able to learn and recognise truncated objects.

## 2   Model

Our purpose is to learn a function $\mathbf{y} = f(\mathbf{x})$, $x \in \mathcal{X}$, $y \in \mathcal{Y}$ which, given an image $\mathbf{x}$, returns the poses $\mathbf{y}$ of the objects portrayed in the image. We use the structured prediction learning framework of [9, 13], which considers along with the input and output variables $\mathbf{x}$ and $\mathbf{y}$, an auxiliary *latent* variable $\mathbf{h} \in \mathcal{H}$ as well (we use $\mathbf{h}$ to specify a refinement to the ground-truth pose parameters). The function $f$ is then defined as $f(\mathbf{x}; w) = \hat{\mathbf{y}}_{\mathbf{x}}(w)$ where

$$(\hat{\mathbf{y}}_{\mathbf{x}}(w), \hat{\mathbf{h}}_{\mathbf{x}}(w)) = \underset{(\mathbf{y},\mathbf{h}) \in \mathcal{Y} \times \mathcal{H}}{\operatorname{argmax}} F(\mathbf{x}, \mathbf{y}, \mathbf{h}; w), \quad F(\mathbf{x}, \mathbf{y}, \mathbf{h}; w) = \langle w, \Psi(\mathbf{x}, \mathbf{y}, \mathbf{h}) \rangle, \qquad (1)$$

and $\Psi(\mathbf{x}, \mathbf{y}, \mathbf{h}) \in \mathbb{R}^d$ is a joint feature map. This maximization estimates both $\mathbf{y}$ and $\mathbf{h}$ from the data $\mathbf{x}$ and corresponds to performing inference. Given training data $(\mathbf{x}_1, \mathbf{y}_1), \ldots, (\mathbf{x}_N, \mathbf{y}_N)$, the parameters $w$ are learned by minimizing the regularized empirical risk

$$R(w) = \frac{1}{2}\|w\|^2 + \frac{C}{N}\sum_{i=1}^{N} \Delta(\mathbf{y}_i, \hat{\mathbf{y}}_i(w), \hat{\mathbf{h}}_i(w)), \quad \text{where} \quad \hat{\mathbf{y}}_i(w) = \hat{\mathbf{y}}_{\mathbf{x}_i}(w), \quad \hat{\mathbf{h}}_i(w) = \hat{\mathbf{h}}_{\mathbf{x}_i}(w).$$
$$(2)$$

Here $\Delta(\mathbf{y}_i, \mathbf{y}, \mathbf{h}) \geq 0$ is an appropriate loss function that encodes the cost of an incorrect prediction.

In this section we develop the model $\Psi(\mathbf{x}, \mathbf{y}, \mathbf{h})$, or equivalently the joint kernel function $K(\mathbf{x}, \mathbf{y}, \mathbf{h}, \mathbf{x}', \mathbf{y}', \mathbf{h}') = \langle \Psi(\mathbf{x}, \mathbf{y}, \mathbf{h}), \Psi(\mathbf{x}', \mathbf{y}', \mathbf{h}') \rangle$, in a number of stages. We first define the kernel for the case of a single unoccluded instance in an image including scale and perturbing transformations, then generalise this to include truncations and aspects; and finally to multiple instances. An appropriate loss function $\Delta(\mathbf{y}_i, \mathbf{y}, \mathbf{h})$ is subsequently defined which takes into account the pose of the object specified by the latent variables.

### 2.1   Restriction and alignment kernel

Denote by $R$ a rectangular region of the image $\mathbf{x}$, and by $\mathbf{x}|_R$ the image cropped to that rectangle. A *restriction kernel* [1] is the kernel $K((\mathbf{x}, R), (\mathbf{x}', R')) = K_{\text{image}}(\mathbf{x}|_R, \mathbf{x}'|_{R'})$ where $K_{\text{image}}$ is an appropriate kernel between images. The goal is that the joint kernel should be large when the two regions have similar appearance.

Our kernel is similar, but captures both the idea of restriction and *alignment*. Let $R_0$ be a reference rectangle and denote by $R(p) = g_p R_0$ the rectangle obtained from $R_0$ by a geometric transformation $g_p : \mathbb{R}^2 \to \mathbb{R}^2$. We call $p$ the *pose* of the rectangle $R(p)$. Let $\bar{\mathbf{x}}$ be an image defined on the reference rectangle $R_0$ and let $H(\bar{\mathbf{x}}) \in \mathbb{R}^d$ be a *descriptor* (e.g. SIFT, HOG, GIST [2]) computed from the image appearance. Then a natural definition of the *restriction and alignment kernel* is

$$K((\mathbf{x}, p), (\mathbf{x}', p')) = K_{\text{descr}}(H(\mathbf{x}; p), H(\mathbf{x}'; p')) \qquad (3)$$

where $K_{\text{descr}}$ is an appropriate kernel for descriptors, and $H(\mathbf{x}; p)$ is the descriptor computed on the transformed patch $\mathbf{x}$ as $H(\mathbf{x}; p) = H(g_p^{-1}\mathbf{x})$.

**Implementation with HOG descriptors.** Our choice of the HOG descriptor puts some limits on the space of poses $p$ that can be efficiently explored. To see this, it is necessary to describe how HOG descriptors are computed.

The computation starts by decomposing the image $\mathbf{x}$ into cells of $d \times d$ pixels (here $d = 8$). The descriptor of a cell is the nine dimensional histogram of the orientation of the image gradient inside the cell (appropriately weighed and normalized as in [2]). We obtain the HOG descriptor of a rectangle of $w \times h$ cells by stacking the enclosed cell descriptors (this is a $9 \times w \times h$ vector). Thus, given the cell histograms, we can immediately obtain the HOG descriptors $H(x, y)$ for all the cell-aligned translations $(x, y)$ of the $dw \times dh$ pixels rectangle. To span rectangles of different scales this construction is simply repeated on the rescaled image $g_s^{-1}\mathbf{x}$, where $g_s(z) = \gamma^s z$ is a rescaling, $\gamma > 0$, and $s$ is a discrete scale parameter.

To further refine pose beyond scale and translation, here we consider an additional perturbation $g_t$, indexed by a pose parameter $t$, selected in a set of transformations $g_1, \ldots, g_T$ (in the experiments we use 16 transformations, obtained from all combinations of rotations of $\pm 5$ and $\pm 10$ degrees and scaling along $x$ of 95%, 90%, 80% and 70%). Those could be achieved in the same manner as scaling by transforming the image $g_t^{-1}\mathbf{x}$ for each one, but this would be very expensive (we would need to recompute the cell descriptors every time). Instead, we approximate such transformations by rearranging the cells of the template (Fig. 1 and 2). First, we partition the $w \times h$ cells of the template into blocks of $m \times m$ cells (e.g. $m = 4$). Then we transform the center of each block according to $g_t$ and we translate the block to the new center (approximated to units of cells). Notice that we could pick $m = 1$ (i.e. move each cell of the template independently), but we prefer to use blocks as this accelerates inference (see Sect. 4).

Hence, pose is for us a tuple $(x, y, s, t)$ representing translation, scale, and additional perturbation. Since HOG descriptors are designed to be compared with a linear kernel, we define

$$K((\mathbf{x}, p), (\mathbf{x}', p')) = \langle \Psi(\mathbf{x}, p), \Psi(\mathbf{x}', p') \rangle, \qquad \Psi(\mathbf{x}, p) = H(\mathbf{x}; p). \tag{4}$$

## 2.2 Modeling truncations

If part of the object is occluded (either by clutter or by the image boundaries), some of the descriptor cells will be either unpredictable or undefined. We explicitly deal with occlusion at the granularity of the HOG cells by adding a field of $w \times h$ binary indicator variables $v \in \{0, 1\}^{wh}$. Here $v_j = 1$ means that the $j$-th cell of the HOG descriptor $H(\mathbf{x}, p)$ should be considered to be visible, and $v_j = 0$ that it is occluded. We thus define a variant of (4) by considering the feature map

$$\Psi(\mathbf{x}, p, v) = \begin{bmatrix} (v \otimes \mathbf{1}_9) \odot H(\mathbf{x}, p) \\ ((\mathbf{1}_{wh} - v) \otimes \mathbf{1}_9) \odot H(\mathbf{x}, p) \end{bmatrix} \tag{5}$$

where $\mathbf{1}_d$ is a $d$-dimensional vector of all ones, $\otimes$ denotes the Kroneker product, and $\odot$ the Hadamard (component wise) product. To understand this expression, recall that $H$ is the stacking of $w \times h$ 9-dimensional histograms, so for instance $(v \otimes \mathbf{1}_9) \odot H(\mathbf{x}, p)$ preserves the visible cells and nulls the others. Eq. (5) is effectively defining a template for the object and one for the occlusions.

Notice that $v$ are in general latent variables and should be estimated as such. However here we note that the most severe and frequent occlusions are caused by the image boundaries (finite field of view). In this case, which we explore in the experiments, we can write $v = v(p)$ as a function of the pose $p$, and remove the explicit dependence on $v$ in $\Psi$. Moreover the truncated HOG cells are undefined and can be assigned a nominal common value. So here we work with a simplified kernel, in which occlusions are represented by a scalar proportional to the number of truncated cells:

$$\Psi(\mathbf{x}, p) = \begin{bmatrix} (v(p) \otimes \mathbf{1}_9) \odot H(\mathbf{x}, p) \\ wh - |v(p)| \end{bmatrix} \tag{6}$$

## 2.3 Modeling aspects

A template model works well as long as pose captures accurately enough the transformations resulting from changes in the viewing conditions. In our model, the pose $p$, combined with the robustness of the HOG descriptor, can absorb a fair amount of viewpoint induced deformation. It cannot, however, capture the 3D structure of a physical object. Therefore, extreme changes of viewpoint require switching between different templates. To this end, we augment pose with an *aspect* indicator $a$ (so that pose is the tuple $p = (x, y, s, t, a)$), which indicates which template to use.

Note that now the concept of pose has been generalized to include a parameter, $a$, which, differently from the others, does not specify a geometric transformation. Nevertheless, pose specifies how the model should be aligned to the image, i.e. by (i) choosing the template that corresponds to the aspect $a$, (ii) translating and scaling such template according to $(x, y, s)$, and (iii) applying to it the additional perturbation $g_t$. In testing, all such parameters are estimated as part of inference. In training, they are initialized from the ground truth data annotations (bounding boxes and aspect labels), and are then refined by estimating the latent variables (Sect. 2.4).

We assign each aspect to a different "slot" of the feature vector $\Psi(\mathbf{x}, p)$. Then we null all but the one of the slots, as indicated by $a$:

$$\Psi(\mathbf{x}, p) = \begin{bmatrix} \delta_{a=1} \Psi_1(\mathbf{x}; p) \\ \vdots \\ \delta_{a=A} \Psi_A(\mathbf{x}; p) \end{bmatrix} \tag{7}$$

where $\Psi_a(\mathbf{x}; p)$ is a feature vector in the form of (6). In this way, we compare different templates for different aspects, as indicated by $a$.

The model can be extended to capture symmetries of the aspects (resulting from symmetries of the objects). For instance, a left view of a bicycle can be obtained by mirroring a right view, so that the same template can be used for both aspects by defining

$$\Psi(\mathbf{x}; p) = \delta_{a=\text{left}} \Psi_{\text{left}}(\mathbf{x}; p) + \delta_{a=\text{right}} \text{ flip } \Psi_{\text{right}}(\mathbf{x}; p), \tag{8}$$

where flip is the operator that "flips" the descriptor (this can be defined in general by computing the descriptor of the mirrored image, but for HOG it reduces to rearranging the descriptor components).

The problem remains of assigning aspects to the training data. In the Pascal VOC data, objects are labeled with one of five aspects: front, left, right, back, undefined. However, such assignments may not be optimal for use in a particular algorithm. Fortunately, our method is able to automatically reassign aspects as part of the estimation of the hidden variables (Sect. 2.4 and Fig. 2).

## 2.4  Latent variables

The PASCAL VOC bounding boxes yield only a coarse estimate of the ground truth pose parameters (e.g. they do not contain any information on the object rotation) and the aspect assignments may also be suboptimal (see previous section). Therefore, we introduce latent variables $\mathbf{h} = (\delta p)$ that encode an adjustment to the ground-truth pose parameters $\mathbf{y} = (p)$. In practice, the adjustment $\delta p$ is a small variation of translation $x, y$, scale $s$, and perturbation $t$, and can switch the aspect $a$ all together.

We modify the feature maps to account for the adjustment in the obvious way. For instance (6) becomes

$$\Psi(\mathbf{x}, p, \delta p) = \begin{bmatrix} (v(p + \delta p) \otimes \mathbf{1}_9) \odot H(\mathbf{x}, p + \delta p) \\ wh - |v(p + \delta p)| \end{bmatrix} \tag{9}$$

## 2.5  Variable number of objects: loss function, bias, training

So far, we have defined the feature map $\Psi(\mathbf{x}, \mathbf{y}) = \Psi(\mathbf{x}; p)$ for the case in which the label $\mathbf{y} = (p)$ contains exactly one object, but an image may contain no or multiple object instances (denoted respectively $\mathbf{y} = \epsilon$ and $\mathbf{y} = (p_1, \ldots, p_n)$). We define the loss function between a ground truth label $\mathbf{y}_i$ and the estimated output $\mathbf{y}$ as

$$\Delta(\mathbf{y}_i, \mathbf{y}) = \begin{cases} 0 & \text{if } \mathbf{y}_i = \mathbf{y} = \epsilon, \\ 1 - \text{overl}(B(p), B(p')) & \text{if } \mathbf{y}_i = (p) \text{ and } \mathbf{y} = (p'), \\ 1 & \text{if } \mathbf{y}_i \neq \epsilon \text{ and } \mathbf{y} = \epsilon, \text{ or } \mathbf{y}_i = \epsilon \text{ and } \mathbf{y} \neq \epsilon, \end{cases} \tag{10}$$

where $B$ is the ground truth bounding box, and $B'$ is the prediction (the smallest axis aligned bounding box that contains the warped template $g_p R_0$). The overlap score between $B$ and $B'$ is given by $\text{overl}(B, B') = |B \cap B'|/|B \cup B'|$. Note that the ground truth poses are defined so that $B(p_l)$ matches the PASCAL provided bounding boxes [1] (or the manually extended ones for the truncated ones).

The hypothesis $\mathbf{y} = \epsilon$ (no object) receives score $F(\mathbf{x}, \epsilon; w) = 0$ by defining $\Psi(\mathbf{x}, \epsilon) = 0$ as in [1]. In this way, the hypothesis $\mathbf{y} = (p)$ is preferred to $\mathbf{y} = \epsilon$ only if $F(\mathbf{x}, p; w) > F(\mathbf{x}, \epsilon; w) = 0$, which implicitly sets the detection threshold to zero. However, there is no reason to assume that this threshold should be appropriate (in Fig. 2 we show that it is not). To learn an arbitrary threshold, it suffices to append to the feature vector $\Psi(\mathbf{x}, p)$ a large constant $\kappa_{\text{bias}}$, so that the score of the hypothesis $\mathbf{y} = (p)$ becomes $F(\mathbf{x}, (p); w) = \langle w, \Psi(\mathbf{x}, p) \rangle + \kappa_{\text{bias}} w_{\text{bias}}$. Note that, since the constant is large, the weight $w_{\text{bias}}$ remains small and has negligible effect on the SVM regularization term.

Finally, an image may contain more than one instance of the object. The model can be extended to this case by setting $F(\mathbf{x}, \mathbf{y}; w) = \sum_{l=1}^{L} F(\mathbf{x}, p_l; w) + R(\mathbf{y})$, where $R(\mathbf{y})$ encodes a "repulsive" force that prevents multiple overlapping detections of the same object. Performing inference with such a model becomes however combinatorial and in general very difficult. Fortunately, in training the problem can be avoided entirely. If an image contains multiple instances, the image is added to the training set multiple times, each time "activating" one of the instances, and "deactivating" the others. Here "deactivating" an instance simply means removing it from the detector search space. Formally, let $p_0$ be the pose of the active instance and $p_1, \ldots, p_N$ the poses of the inactive ones. A pose $p$ is removed from the search space if, and only if, $\max_i \mathrm{overl}(B(p), B(p_i)) \geq \max\{\mathrm{overl}(B(p), B(p_0)), 0.2\}$.

## 3  Optimisation

Minimising the regularised risk $R(w)$ as defined by Eq. (2) is difficult as the loss depends on $w$ through $\hat{\mathbf{y}}_i(w)$ and $\hat{\mathbf{h}}_i(w)$ (see Eq. (1)). It is however possible to optimise an upper bound (derived below) given by

$$\frac{1}{2}\|w\|^2 + \frac{C}{N}\sum_{i=1}^{N} \max_{(\mathbf{y},\mathbf{h})\in\mathcal{Y}\times\mathcal{H}} \Delta(\mathbf{y}_i, \mathbf{y}, \mathbf{h})\left[1 + \langle w, \Psi(\mathbf{x}_i, \mathbf{y}, \mathbf{h})\rangle - \langle w, \Psi(\mathbf{x}_i, \mathbf{y}_i, \mathbf{h}_i^*(w))\rangle\right]. \quad (11)$$

Here $\mathbf{h}_i^*(w) = \mathrm{argmax}_{\mathbf{h}\in\mathcal{H}}\langle w, \Psi(\mathbf{x}_i, \mathbf{y}_i, \mathbf{h})\rangle$ *completes* the label $(\mathbf{y}_i, \mathbf{h}_i^*(w))$ of the sample $\mathbf{x}_i$ (of which only the observed part $\mathbf{y}_i$ is known from the ground truth).

**Alternation optimization.** Eq. (11) is not a convex energy function due to the dependency of $\mathbf{h}_i^*(w)$ on $w$. Similarly to [13], however, it is possible to find a local minimum by alternating optimizing $w$ and estimating $\mathbf{h}_i^*$. To do this, the CCCP algorithm proposed in [13] for the case of margin rescaling, must be extended to the slack rescaling formulation used here.

Starting from an estimate $w_t$ of the solution, define $\mathbf{h}_{it}^* = \mathbf{h}_i(w_t)$, so that, for any $w$,

$$\langle w, \Psi(\mathbf{x}_i, \mathbf{y}_i, \mathbf{h}_i^*(w))\rangle = \max_{\mathbf{h}'}\langle w, \Psi(\mathbf{x}_i, \mathbf{y}_i, \mathbf{h}')\rangle \geq \langle w, \Psi(\mathbf{x}_i, \mathbf{y}_i, \mathbf{h}_{it}^*)\rangle$$

and *the equality holds for $w = w_t$*. Hence the energy (11) is bounded by

$$\frac{1}{2}\|w\|^2 + \frac{C}{N}\sum_{i=1}^{N} \max_{(\mathbf{y},\mathbf{h})\in\mathcal{Y}\times\mathcal{H}} \Delta(\mathbf{y}_i, \mathbf{y}, \mathbf{h})\left[1 + \langle w, \Psi(\mathbf{x}_i, \mathbf{y}, \mathbf{h})\rangle - \langle w, \Psi(\mathbf{x}_i, \mathbf{y}_i, \mathbf{h}_{it}^*)\rangle\right] \quad (12)$$

and the bound is strict for $w = w_t$. Optimising (12) will therefore result in an improvement of the energy (11) as well. The latter can be carried out with the cutting-plane technique of [9].

**Derivation of the bound** (11). The derivation involves a sequence of bounds, starting from

$$\Delta(\mathbf{y}_i, \hat{\mathbf{y}}_i(w), \hat{\mathbf{h}}_i(w)) \leq \Delta(\mathbf{y}_i, \hat{\mathbf{y}}_i(w), \hat{\mathbf{h}}_i(w))\left[1 + \langle w, \Psi(\mathbf{x}_i, \hat{\mathbf{y}}_i(w), \hat{\mathbf{h}}_i(w))\rangle - \langle w, \Psi(\mathbf{x}_i, \mathbf{y}_i, \mathbf{h}_i^*(w))\rangle\right]$$
(13)

This bound holds because, by construction, the quantity in the square brackets is not smaller than one, as can be verified by substituting the definitions of $\hat{\mathbf{y}}_i(w)$, $\hat{\mathbf{h}}_i(w)$ and $\mathbf{h}_i^*(w)$. We further upper bound the loss by

$$\Delta(\mathbf{y}_i, \hat{\mathbf{y}}_i(w), \hat{\mathbf{h}}_i(w)) \leq \Delta(\mathbf{y}_i, \mathbf{y}, \mathbf{h})\left[1 + \langle w, \Psi(\mathbf{x}_i, \mathbf{y}, \mathbf{h})\rangle - \langle w, \Psi(\mathbf{x}_i, \mathbf{y}_i, \mathbf{h}_i^*(w))\rangle\right]\Big|_{\mathbf{y}=\hat{\mathbf{y}}_i(w), \mathbf{h}=\hat{\mathbf{h}}_i(w)}$$
$$\leq \max_{(\mathbf{y},\mathbf{h})\in\mathcal{Y}\times\mathcal{H}} \Delta(\mathbf{y}_i, \mathbf{y}, \mathbf{h})\left[1 + \langle w, \Psi(\mathbf{x}_i, \mathbf{y}, \mathbf{h})\rangle - \langle w, \Psi(\mathbf{x}_i, \mathbf{y}_i, \mathbf{h}_i^*(w))\rangle\right]$$
(14)

Substituting this bound into (2) yields (11). Note that $\hat{\mathbf{y}}_i(w)$ and $\hat{\mathbf{h}}_i(w)$ are defined as the maximiser of $\langle w, \Psi(\mathbf{x}_i, \mathbf{y}, \mathbf{h})\rangle$ alone (see Eq. 1), while the energy maximised in (14) depends on the loss $\Delta(\mathbf{y}_i, \mathbf{y}, \mathbf{h})$ as well.

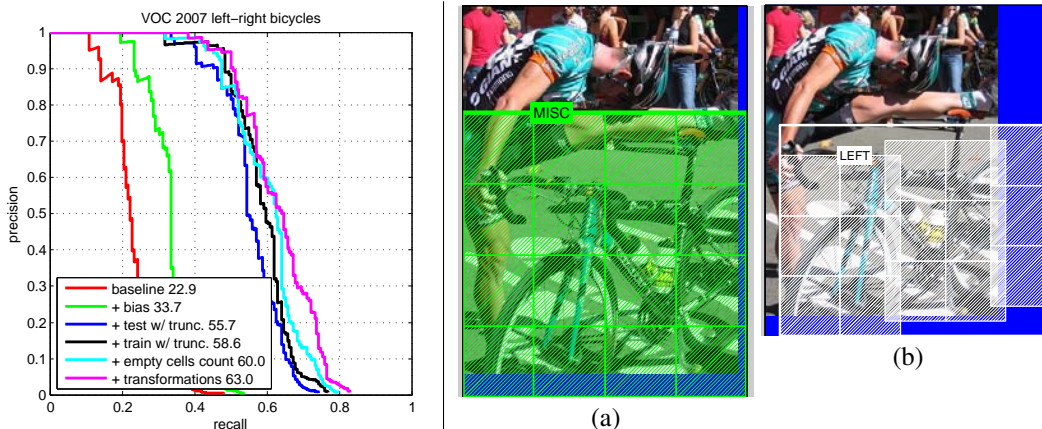

Figure 2: **Effect of different model components.** The left panel evaluates the effect of different components of the model on the task of learning a detector for the left-right facing PASCAL VOC 2007 bicycles. In order of increasing AP (see legend): baseline model (see text); bias term (Sect. 2.5); detecting trunctated instances, training on truncated instances, and counting the truncated cells as a feature (Sect.: 2.2); with searching over small translation, scaling, rotation, skew (Sect. 2.1). Right panel: (a) Original VOC specified bounding box and aspect; (b) alignment and aspect after pose inference – in addition to translation and scale, our templates are searched over a set of small perturbations. This is implemented efficiently by breaking the template into blocks (dashed boxes) and rearranging those. Note that blocks can partially overlap to capture foreshortening. The ground truth pose parameters are approximate because they are obtained from bounding boxes (a). The algorithm improves their estimate as part of inference of the latent variables $\mathbf{h}$. Notice that not only translation, scale, and small jitters are re-estimated, but also the aspect subclass can be updated. In the example, an instance originally labeled as *misc* (a) is reassigned to the *left* aspect (b).

# 4 Experiments

**Data.** As training data we use the PASCAL VOC annotations. Each object instance is labeled with a bounding box and a categorical aspect variable (left, right, front, back, undefined). From the bounding box we estimate translation and scale of the object, and we use aspect to select one of multiple HOG templates. Symmetric aspects (e.g. left and right) are mapped to the same HOG template as suggested in Sect. 2.3.

While our model is capable of handling correctly truncations, truncated bounding boxes provide a poor estimate of the pose of the object pose which prevents using such objects for training. While we could simply avoid training with truncated boxes (or generate artificially truncated examples whose pose would be known), we prefer exploiting all the available training data. To do this, we manually augment all truncated PASCAL VOC annotations with an additional "physical" bounding box. The purpose is to provide a better initial guess for the object pose, which is then refined by optimizing over the latent variables.

**Training and testing speed.** Performing inference with the model requires evaluating $\langle w, \Psi(\mathbf{x}, p) \rangle$ for all possible poses $p$. This means matching a HOG template $O(WHTA)$ times, where $W \times H$ is the dimension of the image in cells, $T$ the number of perturbations (Sect. 2.1), and $A$ the number of aspects (Sect. 2.3).[1] For a given scale and aspect, matching the template for all locations reduces to convolution. Moreover, by breaking the template into blocks (Fig. 2) and pre-computing the convolution with each of those, we can quickly compute perturbations of the template. All in all, detection requires roughly 30 seconds per image with the full model and four aspects. The cutting plane algorithm used to minimize (12) requires at each iteration solving problems similar to inference. This can be easily parallelized, greatly improving training speed. To detect additional objects at test time we run inference multiple times, but excluding all detections that overlap by more than 20% with any previously detected object.

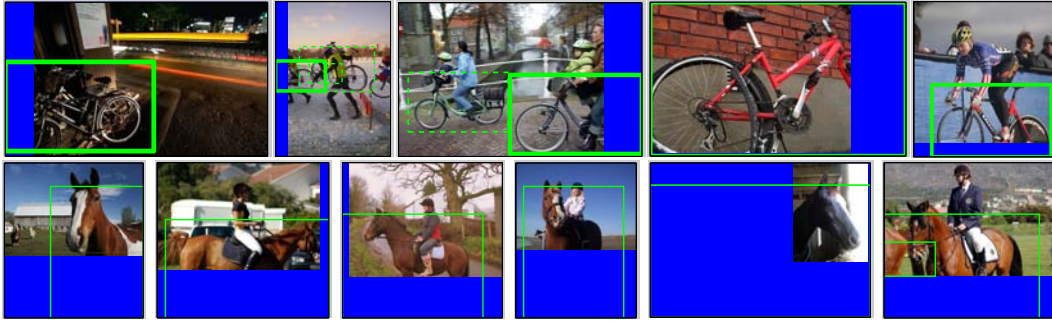

Figure 3: **Top row.** Examples of detected bicycles. The dashed boxes are bicycles that were detected with or without truncation support, while the solid ones were detectable only when truncations were considered explicitly. **Bottom row.** Some cases of correct detections despite extreme truncation for the horse class.

**Benefit of various model components.** Fig. 2 shows how the model improves by the successive introduction of the various features of the model. The example is carried on the VOC left-right facing bicycle, but similar effects were observed for other categories. The baseline model uses only the HOG template without bias, truncations, nor pose refinement (Sect. 2.1). The two most significant improvements are (a) the ability of detecting truncated instances (+22% AP, Fig. 3) and (b) the addition of the bias (+11% AP). Training with the truncated instances, adding the number of occluded HOG cells as a feature component, and adding jitters beyond translation and scaling all yield an improvement of about +2–3% AP.

**Full model.** The model was trained to detect the class bicycle in the PASCAL VOC 2007 data, using five templates, initialized from the PASCAL labeling left, right, front/rear, other. Initially, the pose refinement $\mathbf{h}$ is null and the alternation optimization algorithm is iterated five times to estimate the model $w$ and refinement $\mathbf{h}$. The detector is then tested on all the test data, enabling multiple detections per image, and computing average-precision as specified by [3]. The AP score was 47%. By comparison, the state of the art for this category [8] achieves 56%. The experiment was repeated for the class horse, obtaining a score of 40%. By comparison, the state of the art on this category, our MKL sliding window classifier [10], achieves 51%. Note that the proposed method uses only HOG, while the others use a combination of at least two features. However [4], using only HOG but a flexible part model, also achieves superior results. Further experiments are needed to evaluate the combined benefits of truncation/occlusion handling (proposed here), with multiple features [10] and flexible parts [4].

## Conclusions

We have shown how structured output regression with latent variables provides an integrated and effective solution to many problems in object detection: truncations, pose variability, multiple objects, and multiple aspects can all be dealt in a consistent framework.

While we have shown that truncated examples can be used for training, we had to manually extend the PASCAL VOC annotations for these cases to include rough "physical" bounding boxes (as a hint for the initial pose parameters). We plan to further extend the approach to infer pose for truncated examples in a fully automatic fashion (weak supervision).

**Acknowledgments.** We are grateful for discussions with Matthew Blaschko. Funding was provided by the EU under ERC grant VisRec no. 228180; the RAEng, Microsoft, and ONR MURI N00014-07-1-0182.

## Footnotes

[1] Note that we do not multiply by the number $S$ of scales as at each successive scale $W$ and $H$ are reduced geometrically.

# References

[1] M. B. Blaschko and C. H. Lampert. Learning to localize objects with structured output regression. In *Proc. ECCV*, 2008.

[2] N. Dalal and B. Triggs. Histograms of oriented gradients for human detection. In *Proc. CVPR*, 2005.

[3] M. Everingham, L. Van Gool, C. K. I. Williams, J. Winn, and A. Zisserman. The PASCAL Visual Object Classes Challenge 2008 (VOC2008) Results. `http://www.pascal-network.org/challenges/VOC/voc2008/workshop/index.html`, 2008.

[4] P. F. Felzenszwalb, R. B. Grishick, D. McAllister, and D. Ramanan. Object detection with discriminatively trained part based models. *PAMI*, 2009.

[5] R. Fergus, P. Perona, and A. Zisserman. Object class recognition by unsupervised scale-invariant learning. In *Proceedings of the IEEE Conference on Computer Vision and Pattern Recognition*, volume 2, pages 264–271, June 2003.

[6] K. Hotta. Robust face detection under partial occlusion. In *Proceedings of the IEEE International Conference on Image Processing*, 2004.

[7] Y. Y. Lin, T. L. Liu, and C. S. Fuh. Fast object detection with occlusions. In *Proceedings of the European Conference on Computer Vision*, pages 402–413. Springer-Verlag, May 2004.

[8] P. Schnitzspan, M. Fritz, S. Roth, and B. Schiele. Discriminative structure learning of hierarchical representations for object detection. In *Proc. CVPR*, 2009.

[9] I. Tsochantaridis, T. Hofmann, T. Joachims, and Y. Altun. Support vector machine learning for interdependent and structured output spaces. In *Proc. ICML*, 2004.

[10] A. Vedaldi, V. Gulshan, M. Varma, and A. Zisserman. Multiple kernels for object detection. In *Proc. ICCV*, 2009.

[11] O. Williams, A. Blake, and R. Cipolla. The variational ising classifier (VIC) algorithm for coherently contaminated data. In *Proc. NIPS*, 2005.

[12] J. Winn and J. Shotton. The Layout Consistent Random Field for Recognizing and Segmenting Partially Occluded Objects. In *Proc. CVPR*, 2006.

[13] C.-N. J. Yu and T. Joachims. Learning structural SVMs with latent variables. In *Proc. ICML*, 2009.

